# Label Embedding Trees for Large Multi-Class Tasks

**Samy Bengio**[1]     **Jason Weston**[1]     **David Grangier**[2]

[1] Google Research, New York, NY
{bengio, jweston}@google.com
[2]NEC Labs America, Princeton, NJ
{dgrangier}@nec-labs.com

## Abstract

Multi-class classification becomes challenging at test time when the number of classes is very large and testing against every possible class can become computationally infeasible. This problem can be alleviated by imposing (or learning) a structure over the set of classes. We propose an algorithm for learning a tree-structure of classifiers which, by optimizing the overall tree loss, provides superior accuracy to existing tree labeling methods. We also propose a method that learns to embed labels in a low dimensional space that is faster than non-embedding approaches and has superior accuracy to existing embedding approaches. Finally we combine the two ideas resulting in the *label embedding tree* that outperforms alternative methods including One-vs-Rest while being orders of magnitude faster.

## 1   Introduction

Datasets available for prediction tasks are growing over time, resulting in increasing scale in all their measurable dimensions: separate from the issue of the growing number of examples $m$ and features $d$, they are also growing in the number of classes $k$. Current multi-class applications such as web advertising [6], textual document categorization [11] or image annotation [12] have tens or hundreds of thousands of classes, and these datasets are *still growing*. This evolution is challenging traditional approaches [1] whose test time grows at least linearly with $k$.

At training time, a practical constraint is that learning should be feasible, i.e. it should not take more than a few days, and must work with the memory and disk space requirements of the available hardware. Most algorithms' training time, at best, linearly increases with $m$, $d$ and $k$; algorithms that are quadratic or worse with respect to $m$ or $d$ are usually discarded by practitioners working on real large scale tasks. At testing time, depending on the application, very specific time constraints are necessary, usually measured in milliseconds, for example when a real-time response is required or a large number of records need to be processed. Moreover, memory usage restrictions may also apply. Classical approaches such as One-vs-Rest are at least $O(kd)$ in both speed (of testing a single example) and memory. This is prohibitive for large scale problems [6, 12, 26].

In this work, we focus on algorithms that have a classification speed sublinear at testing time in $k$ as well as having limited dependence on $d$ with best-case complexity $O(d_e(\log k + d))$ with $d_e \ll d$ and $d_e \ll k$. In experiments we observe no loss in accuracy compared to methods that are $O(kd)$, further, memory consumption is reduced from $O(kd)$ to $O(kd_e)$. Our approach rests on two main ideas: firstly, an algorithm for *learning* a *label tree*: each node makes a prediction of the subset of labels to be considered by its children, thus decreasing the number of labels $k$ at a logarithmic rate until a prediction is reached. We provide a novel algorithm that both learns the sets of labels at each node, and the predictors at the nodes to optimize the overall tree loss, and show that this approach is superior to existing tree-based approaches [7, 6] which typically lose accuracy compared to $O(kd)$ approaches. Balanced label trees have $O(d \log k)$ complexity as the predictor at each node is still

---
**Algorithm 1** Label Tree Prediction Algorithm
---
    **Input:** test example x, parameters $T$.
    Let $s = 0$.                                            *- Start at the root node*
    **repeat**
        Let $s = \operatorname{argmax}_{\{c:(s,c)\in E\}} f_c(x)$.         *- Traverse to the most confident child.*
    **until** $|\ell_s| = 1$                            *- Until this uniquely defines a single label.*
    Return $\ell_s$.
---

linear in $d$. Our second main idea is to learn an embedding of the labels into a space of dimension $d_e$ that again still optimizes the overall tree loss. Hence, we are required at test time to: (1) map the test example in the *label embedding* space with cost $O(dd_e)$ and then (2) predict using the label tree resulting in our overall cost $O(d_e(\log k + d))$. We also show that our label embedding approach outperforms other recently proposed label embedding approaches such as compressed sensing [17].

The rest of the paper is organized as follows. Label trees are discussed and label tree learning algorithms are proposed in Section 2. Label embeddings are presented in Section 3. Related prior work is presented in Section 4. An experimental study on three large tasks is given in Section 5 showing the good performance of our proposed techniques. Finally, Section 6 concludes.

## 2   Label Trees

A label tree is a tree $T = (N, E, F, L)$ with $n+1$ indexed nodes $N = \{0, \ldots n\}$, a set of edges $E = \{(p_1, c_1), (p_{|E|}, c_{|E|})\}$ which are ordered pairs of parent and child node indices, *label predictors* $F = \{f_1, \ldots, f_n\}$ and *label sets* $L = \{\ell_0, \ldots, \ell_n\}$ associated to each node. The root node is labeled with index 0. The edges $E$ are such that all other nodes have one parent, but they can have an arbitrary number of children (but still in all cases $|E| = n$). The label sets indicate the set of labels to which a point should belong if it arrives at the given node, and progress from generic to specific along the tree, i.e. the root label set contains all classes $|\ell_0| = k$ and each child label set is a subset of its parent label set with $\ell_p = \bigcup_{(p,c)\in E} \ell_c$. We differentiate between *disjoint* label trees where there are only $k$ leaf nodes, one per class, and hence any two nodes $i$ and $j$ at the same depth cannot share any labels, $\ell_i \cap \ell_j = \emptyset$, and *joint* label trees that can have more than $k$ leaf nodes.

Classifying an example with the label tree is achieved by applying Algorithm 1. Prediction begins at the root node ($s = 0$) and for each edge leading to a child $(s, c) \in E$ one computes the score of the label predictor $f_c(x)$ which predicts whether the example $x$ belongs to the set of labels $\ell_c$. One takes the most confident prediction, traverses to that child node, and then repeats the process. Classification is complete when one arrives at a node that identifies only a single label, which is the predicted class.

Instances of label trees have been used in the literature before with various methods for choosing the parameters $(N, E, F, L)$. Due to the difficulty of learning, many methods make approximations such as a random choice of $E$ and optimization of $F$ that does not take into account the overall loss of the entire system leading to suboptimal performance (see [7] for a discussion). Our goal is to provide an algorithm to learn these parameters to optimize the overall empirical loss (called the *tree loss*) as accurately as possible for a given tree size (speed).

We can define the tree loss we wish to minimize as:

$$R(f_{tree}) = \int I(f_{tree}(x) \neq y) dP(x, y) = \int \max_{i \in B(x) = \{b_1(x), \ldots b_{D(x)}(x)\}} I(y \notin \ell_i) dP(x, y) \quad (1)$$

where $I$ is the indicator function and

$$b_j(x) = \operatorname{argmax}_{\{c \,:\, (b_{j-1}(x), c) \in E\}} f_c(x)$$

is the index of the winning ("best") node at depth $j$, $b_0(x) = 0$, and $D(x)$ is the depth in the tree of the final prediction for $x$, i.e. the number of loops plus one of the repeat block when running Algorithm 1. The tree loss measures an intermediate loss of 1 for each prediction at each depth $j$ of the label tree where the true label is not in the label set $\ell_{b_j(x)}$. The final loss for a single example is the $\max$ over these losses, because if any one of these classifiers makes a mistake then regardless

of the other predictions the wrong class will still be predicted. Hence, any algorithm wishing to optimize the overall tree loss should train all the nodes *jointly* with respect to this maximum.

We will now describe how we propose to learn the parameters $T$ of our label tree. In the next subsection we show how to minimize the tree loss for a given fixed tree ($N$, $E$ and $L$ are fixed, $F$ is to be learned). In the following subsection, we will describe our algorithm for learning $N$, $E$ and $L$.

## 2.1 Learning with a Fixed Label Tree

Let us suppose we are given a fixed label tree $N, E, L$ chosen in advance. Our goal is simply to minimize the tree loss (1) over the variables $F$, given training data $\{(x_i, y_i)\}_{i=1,\ldots,m}$. We follow the standard approach of minimizing the empirical loss over the data, while regularizing our solution. We consider two possible algorithms for solving this problem.

**Relaxation 1: Independent convex problems** The simplest (and poorest) procedure is to consider the following relaxation to this problem:

$$R_{emp}(f_{tree}) = \frac{1}{m}\sum_{i=1}^{m}\max_{j\in B(x)} I(y_i \notin \ell_j) \ \leq \ \frac{1}{m}\sum_{i=1}^{m}\sum_{j=1}^{n} I(\mathrm{sgn}(f_j(x_i)) = \mathcal{C}_j(y_i))$$

where $\mathcal{C}_j(y) = 1$ if $y \in \ell_j$ and -1 otherwise. The number of errors counted by the approximation cannot be less than the *empirical tree loss* $R_{emp}$ as when, for a particular example, the loss is zero for the approximation it is also zero for $R_{emp}$. However, the approximation can be much larger because of the sum.

One then further approximates this by replacing the indicator function with the hinge loss and choosing linear (or kernel) models of the form $f_i(x) = w_i^\top \phi(x)$. We are then left with the following convex problem: minimize

$$\sum_{j=1}^{n}\left(\gamma||w_j||^2 + \frac{1}{m}\sum_{i=1}^{m}\xi_{ij}\right) \text{ s.t. } \forall i,j, \ \left\{ \begin{array}{l} \mathcal{C}_j(y_i)f_j(x_i) \geq 1 - \xi_{ij} \\ \xi_{ij} \geq 0 \end{array} \right.$$

where we also added a classical 2-norm regularizer controlled by the hyperparameter $\gamma$. In fact, this can be split into $n$ independent convex problems because the hyperplanes $w_i$, $i = 1, \ldots, n$, do not interact in the objective function. We consider this simple relaxation as a baseline approach.

**Relaxation 2: Tree Loss Optimization (Joint convex problem)** We propose a tighter minimization of the tree loss with the following:

$$\frac{1}{m}\sum_{i=1}^{m}\xi_i^\alpha$$

$$\text{s.t. } f_r(x_i) \geq f_s(x_i) - \xi_i, \ \ \forall r,s : y_i \in \ell_r \wedge y_i \notin \ell_s \wedge (\exists p : (p,r) \in E \wedge (p,s) \in E) \quad (2)$$

$$\xi_i \geq 0, \ \ i = 1, \ldots, m. \quad (3)$$

When $\alpha$ is close to zero, the shared slack variables simply count a single error if any of the predictions at any depth of the tree are incorrect, so this is very close to the true optimization of the tree loss. This is measured by checking, out of all the nodes that share the same parent, if the one containing the true label in its label set is highest ranked. In practice we set $\alpha = 1$ and arrive at a convex optimization problem. Nevertheless, unlike relaxation (1) the max is not approximated with a sum. Again, using the hinge loss and a 2-norm regularizer, we arrive at our final optimization problem:

$$\gamma\sum_{j=1}^{n}||w_j||^2 + \frac{1}{m}\sum_{i=1}^{m}\xi_i \quad (4)$$

subject to constraints (2) and (3).

## 2.2 Learning Label Tree Structures

The previous section shows how to optimize the label predictors $F$ while the nodes $N$, edges $E$ and label sets $L$ which specify the structure of the tree are fixed in advance. However, we want to be able to learn specific tree structures dependent on our prediction problem such that we minimize the

---

**Algorithm 2** Learning the Label Tree Structure

---

**Train** $k$ One-vs-Rest classifiers $\bar{f}_1, \ldots, \bar{f}_k$ independently (no tree structure is used).

**Compute** the confusion matrix $\bar{C}_{ij} = |\{(x, y_i) \in \mathcal{V} : \operatorname{argmax}_r \bar{f}_r(x) = j\}|$ on validation set $\mathcal{V}$.

**For each** internal node $l$ of the tree, from root to leaf, partition its label set $\ell_l$ between its children's label sets $L_l = \{\ell_c : c \in N_l\}$, where $N_l = \{c \in N \ : \ (l, c) \in E\}$ and $\cup_{c \in N_l} \ell_c = \ell_l$, by maximizing:

$$R_l(L_l) = \sum_{c \in N_l} \sum_{y_p, y_q \in \ell_c} A_{pq}, \quad \text{where } A = \frac{1}{2}(\bar{C} + \bar{C}^\top) \text{ is the symmetrized confusion matrix,}$$

subject to constraints preventing trivial solutions, e.g. putting all labels in one set (see [4]).

This optimization problem (including the appropriate constraints) is a graph cut problem and it can be solved with standard spectral clustering, i.e. we use $A$ as the affinity matrix for step 1 of the algorithm given in [21], and then apply all of its other steps (2-6).

**Learn** the parameters $f$ of the tree by minimizing (4) subject to constraints (2) and (3).

---

overall tree loss. This section describes an algorithm for learning the parameters $N$, $E$ and $L$, i.e. optimizing equation (1) with respect to these parameters.

The key to the generalization ability of a particular choice of tree structure is the learnability of the label sets $\ell$. If some classes are often confused but are in different label sets the functions $f$ may not be easily learnable, and the overall tree loss will hence be poor. For example for an image labeling task, a decision in the tree between two label sets, one containing tiger and jaguar labels versus one containing frog and toad labels is presumably more learnable than (tiger, frog) vs. (jaguar, toad).

In the following, we consider a learning strategy for disjoint label trees (the methods in the previous section were for both joint and disjoint trees). We begin by noticing that $R_{emp}$ can be rewritten as:

$$R_{emp}(f_{tree}) = \frac{1}{m} \sum_{i=1}^{m} \max_j \left( I(y_i \in \ell_j) \sum_{\bar{y} \notin \ell_j} C(x_i, \bar{y}) \right)$$

where $C(x_i, \bar{y}) = I(f_{tree}(x_i) = \bar{y})$ is the confusion of labeling example $x_i$ (with true label $y_i$) with label $\bar{y}$ instead. That is, the tree loss for a given example is 1 if there is a node $j$ in the tree containing $y_i$, but we predict a different node at the same depth leading to a prediction not in the label set of $j$.

Intuitively, the confusion of predicting node $i$ instead of $j$ comes about because of the class confusion between the labels $y \in \ell_i$ and the labels $\bar{y} \in \ell_j$. Hence, to provide the smallest tree loss we want to group together labels into the same label set that are likely to be confused at test time. Unfortunately we do not know the confusion matrix of a particular tree without training it first, but as a proxy we can use the class confusion matrix of a surrogate classifier with the supposition that the matrices will be highly correlated. This motivates the proposed Algorithm 2. The main idea is to recursively partition the labels into label sets between which there is little confusion (measuring confusion using One-vs-Rest as a surrogate classifier) solving at each step a graph cut problem where standard spectral clustering is applied [20, 21]. The objective function of spectral clustering penalizes unbalanced partitions, hence encouraging balanced trees. (To obtain logarithmic speedups the tree has to be balanced; one could also enforce this constraint directly in the $k$-means step.)

The results in Section 5 show that our learnt trees outperform random structures and in fact match the accuracy of not using a tree at all, while being orders of magnitude faster.

## 3 Label Embeddings

An orthogonal angle of attack of the solution of large multi-class problems is to employ *shared* representations for the labelings, which we term label embeddings. Introducing the function $\phi(y) = (0, \ldots, 0, 1, 0, \ldots, 0)$ which is a $k$-dimensional vector with a 1 in the $y^{th}$ position and 0 otherwise, we would like to find a linear embedding $\mathcal{E}(y) = V\phi(y)$ where $V$ is a $d_e \times k$ matrix assuming that labels $y \in \{1, \ldots, k\}$. Without a tree structure, multi-class classification is then achieved with:

$$f_{embed}(x) = \operatorname{argmax}_{i=1,\ldots,k} S\left(Wx, V\phi(i)\right) \tag{5}$$

where $W$ is a $d_e \times d$ matrix of parameters and $S(\cdot, \cdot)$ is a measure of similarity, e.g. an inner product or negative Euclidean distance. This method, unlike label trees, is unfortunately still linear with respect to $k$. However, it does have better behavior with respect to the feature dimension $d$, with $O(d_e(d + k))$ testing time, compared to methods such as One-vs-Rest which is $O(kd)$. If the embedding dimension $d_e$ is much smaller than $d$ this gives a significant saving.

There are several ways we could train such models. For example, the method of compressed sensing [17] has a similar form to (5), but the matrix $V$ is not learnt but chosen randomly, and only $W$ is learnt. In the next section we will show how we can train such models so that the matrix $V$ captures the semantic similarity between classes, which can improve generalization performance over random choices of $V$ in an analogous way to the improvement of label trees over random trees. Subsequently, we will show how to combine label embeddings with label trees to gain the advantages of both approaches.

### 3.1 Learning Label Embeddings (Without a Tree)

We consider two possibilities for learning $V$ and $W$.

**Sequence of Convex Problems** Firstly, we consider learning the label embedding by solving a sequence of convex problems using the following method. First, train independent (convex) classifiers $\bar{f}_i(x)$ for each class $1, \ldots, k$ and compute the $k \times k$ confusion matrix $\bar{C}$ over the data $(x_i, y_i)$, i.e. the same as the first two steps of Algorithm 2. Then, find the label embedding vectors $V_i$ that minimize:

$$\sum_{i,j=1}^{k} A_{ij}||V_i - V_j||^2, \quad \text{where } A = \frac{1}{2}(\bar{C} + \bar{C}^\top) \text{ is the symmetrized confusion matrix,}$$

subject to the constraint $V^\top D V = I$ where $D_{ii} = \sum_j A_{ij}$ (to prevent trivial solutions) which is the same problem solved by Laplacian Eigenmaps [4]. We then obtain an embedding matrix $V$ where similar classes $i$ and $j$ should have small distance between their vectors $V_i$ and $V_j$. All that remains is to learn the parameters $W$ of our model. To do this, we can then train a convex multi-class classifier utilizing the label embedding $V$: minimize

$$\gamma||W||_{FRO} + \frac{1}{m}\sum_{i=1}^{m} \xi_i$$

where $||.||_{FRO}$ is the Frobenius norm, subject to constraints:

$$||Wx_i - V\phi(i)||^2 \leq ||Wx_i - V\phi(j)||^2 + \xi_i, \quad \forall j \neq i \tag{6}$$

$$\xi_i \geq 0, \quad i = 1, \ldots, m.$$

Note that the constraint (6) is linear as we can multiply out and subtract $||Wx_i||^2$ from both sides. At test time we employ equation (5) with $S(z, z') = -||z - z'||$.

**Non-Convex Joint Optimization** The second method is to learn $W$ and $V$ jointly, which requires non-convex optimization. In that case we wish to directly minimize:

$$\gamma||W||_{FRO} + \frac{1}{m}\sum_{i=1}^{m} \xi_i$$

$$\text{subject to} \quad (Wx_i)^\top V\phi(i) \geq (Wx_i)^\top V\phi(j) - \xi_i, \quad \forall j \neq i$$

and $||V_i|| \leq 1$, $\xi_i \geq 0$, $i = 1, \ldots, m$. We optimize this using stochastic gradient descent (with randomly initialized weights) [8]. At test time we employ equation (5) with $S(z, z') = z^\top z'$.

### 3.2 Learning Label Embedding Trees

In this work, we also propose to combine the use of embeddings and label trees to obtain the advantages of both approaches, which we call the *label embedding tree*. At test time, the resulting label embedding tree prediction is given in Algorithm 3. The label embedding tree has potentially $O(d_e(d + log(k)))$ testing speed, depending on the structure of the tree (e.g. being balanced).

**Algorithm 3** Label Embedding Tree Prediction Algorithm

---

  **Input:** test example x, parameters $T$.
  Compute $z = Wx$.                                                      *- Cache prediction on example*
  Let $s = 0$.                                                     *- Start at the root node*
  **repeat**                                            *- Traverse to the most*
     Let $s = \mathrm{argmax}_{\{c:(s,c)\in E\}} f_c(x) = \mathrm{argmax}_{\{c:(s,c)\in E\}} z^\top \mathcal{E}(c)$.      *confident child.*
  **until** $|\ell_s| = 1$                      *- Until this uniquely defines a single label.*
  Return $\ell_s$.

---

To learn a label embedding tree we propose the following minimization problem:

$$\gamma ||W||_{FRO} + \frac{1}{m}\sum_{i=1}^{m}\xi_i$$

subject to constraints:

$$(Wx_i)^\top V\phi(r) \geq (Wx_i)^\top V\phi(s) - \xi_i, \ \ \forall r,s : y_i \in \ell_r \wedge y_i \notin \ell_s \wedge (\exists p : (p,r) \in E \wedge (p,s) \in E)$$

$$||V_i|| \leq 1, \ \ \xi_i \geq 0, \ \ i = 1,\ldots,m.$$

This is essentially a combination of the optimization problems defined in the previous two Sections. Learning the tree structure for these models can still be achieved using Algorithm 2.

## 4 Related Work

**Multi-class classification** is a well studied problem. Most of the prior approaches build upon binary classification and have a classification cost which grows at least linearly with the number of classes $k$. Common multi-class strategies include one-versus-rest, one-versus-one, label ranking and Decision Directed Acyclic Graph (DDAG). One-versus-rest [25] trains $k$ binary classifiers discriminating each class against the rest and predicts the class whose classifier is the most confident, which yields a linear testing cost $O(k)$. One-versus-one [16] trains a binary classifier for each pair of classes and predicts the class getting the most pairwise preferences, which yields a quadratic testing cost $O(k \cdot (k-1)/2)$. Label ranking [10] learns to assign a score to each class so that the correct class should get the highest score, which yields a linear testing cost $O(k)$. DDAG [23] considers the same $k \cdot (k-1)/2$ classifiers as one-versus-one but achieves a linear testing cost $O(k)$. All these methods are reported to perform similarly in terms of accuracy [25, 23].

Only a few prior techniques achieve sub-linear testing cost. One way is to simply remove labels the classifier performs poorly on [11]. Error correcting code approaches [13] on the other hand represent each class with a binary code and learn a binary classifier to predict each bit. This means that the testing cost could potentially be $O(\log k)$. However, in practice, these approaches need larger redundant codes to reach competitive performance levels [19]. Decision trees, such as C4.5 [24], can also yield a tree whose depth (and hence test cost) is logarithmic in $k$. However, testing complexity also grows linearly with the number of training examples making these methods impractical for large datasets [22].

Filter tree [7] and Conditional Probability Tree (CPT) [6] are logarithmic approaches that have been introduced recently with motivations similar to ours, i.e. addressing large scale problems with a thousand classes or more. Filter tree considers a random binary tree in which each leaf is associated with a class and each node is associated with a binary classifier. A test example traverses the tree from the root. At each node, the node classifier decides whether the example is directed to the right or to the left subtree, each of which are associated to half of the labels of the parent node. Finally, the label of the reached leaf is predicted. Conditional Probability Tree (CPT) relies on a similar paradigm but builds the tree during training. CPT considers an online setup in which the set of classes is discovered during training. Hence, CPT builds the tree greedily: when a new class is encountered, it is added by splitting an existing leaf. In our case, we consider that the set of classes are available prior to training and propose to tessellate the class label sets such that the node classifiers are likely to achieve high generalization performance. This contribution is shown to have a significant advantage in practice, see Section 5.

Finally, we should mention that a related active area of research involves partitioning the *feature space* rather than the *label space*, e.g. using hierarchical experts [18], hashing [27] and kd-trees [5].

**Label embedding** is another key aspect of our work when it comes to efficiently handling thousands of classes. Recently, [26] proposed to exploit class taxonomies via embeddings by learning to project input vectors and classes into a common space such that the classes close in the taxonomy should have similar representations while, at the same time, examples should be projected close to their class representation. In our case, we do not rely on a pre-existing taxonomy: we also would like to assign similar representations to similar classes but solely relying on the training data. In that respect, our work is closer to work in information retrieval [3], which proposes to embed documents – not classes – for the task of document ranking. Compressed sensing based approaches [17] do propose to embed class labels, but rely on a random projection for embedding the vector representing class memberships, with the added advantages of handling problems for which multiple classes are active for a given example. However, relying on a random projection does not allow for the class embedding to capture the relation between classes. In our experiments, this aspect is shown to be a drawback, see Section 5. Finally, the authors of [2] do propose an embedding approach over class labels, but it is not clear to us if their approach is scalable to our setting.

## 5   Experimental Study

We consider three datasets: one publicly available image annotation dataset and two proprietary datasets based on images and textual descriptions of products.

**ImageNet Dataset** ImageNet [12] is a new image dataset organized according to WordNet [14] where quality-controlled human-verified images are tagged with labels. We consider the task of annotating images from a set of about 16 thousand labels. We split the data into 2.5M images for training, 0.8M for validation and 0.8M for testing, removing duplicates between training, validation and test sets by throwing away test examples which had too close a nearest neighbor training or validation example in feature space. Images in this database were represented by a large but sparse vector of color and texture features, known as visual terms, described in [15].

**Product Datasets** We had access to a large proprietary database of about 0.5M product descriptions. Each product is associated with a textual description, an image, and a label. There are ≈18 thousand unique labels. We consider two tasks: predicting the label given the textual description, and predicting the label given the image. For the text task we extracted the most frequent set of 10 thousand words (discounting stop words) to yield a textual dictionary, and represented each document by a vector of counts of these words in the document, normalized using tf-idf. For the image task, images were represented by a dense vector of 1024 real values of texture and color features.

Table 1 summarizes the various datasets. Next, we describe the approaches that we compared.

**Flat versus Tree Learning Approaches** In Table 2 we compare label tree predictor training methods from Section 2.1: the baseline relaxation 1 ("Independent Optimization") versus our proposed relaxation 2 ("Tree Loss Optimization"), both of which learn the classifiers for fixed trees; and we compare our "Learnt Label Tree" structure learning algorithm from Section 2.2 to random structures. In all cases we considered disjoint trees of depth 2 with 200 internal nodes. The results show that *learnt structure* performs better than *random structure* and *tree loss optimization* is superior to *independent optimization*. We also compare to three other baselines: One-vs-Rest large margin classifiers trained using the passive aggressive algorithm [9], the Filter Tree [7] and the Conditional Probability Tree (CPT) [6]. For all algorithms, hyperparameters are chosen using the validation set. The combination of Learnt Label Tree structure and Tree Loss Optimization for the label predictors is the only method that is comparable to or better than One-vs-Rest while being around $60\times$ faster to compute at test time.

For ImageNet one could wonder how well using WordNet (a graph of human annotated label similarities) to build a tree would perform instead. We constructed a matrix $C$ for Algorithm 2 where $C_{ij} = 1$ if there is an edge in the WordNet graph, and 0 otherwise, and used that to learn a label tree as before, obtaining 0.99% accuracy using "Independent Optimization". This is better than a random tree but not as good as using the confusion matrix, implying that the best tree to use is the one adapted to the supervised task of interest.

Table 1: **Summary Statistics of the Three Datasets Used in the Experiments.**

| Statistics | ImageNet | Product Descriptions | Product Images |
|---|---|---|---|
| Task | image annotation | product categorization | image annotation |
| Number of Training Documents | 2518604 | 417484 | 417484 |
| Number of Test Documents | 839310 | 60278 | 60278 |
| Validation Documents | 837612 | 105572 | 105572 |
| Number of Labels | 15952 | 18489 | 18489 |
| Type of Documents | images | texts | images |
| Type of Features | visual terms | words | dense image features |
| Number of Features | 10000 | 10000 | 1024 |
| Average Feature Sparsity | 97.5% | 99.6% | 0.0% |

Table 2: **Flat versus Tree Learning Results** Test set accuracies for various tree and non-tree methods on three datasets. Speed-ups compared to One-vs-Rest are given in brackets.

| Classifier | Tree Type | ImageNet | Product Desc. | Product Images |
|---|---|---|---|---|
| One-vs-Rest | None (flat) | 2.27% [1×] | 37.0% [1×] | 12.6% [1×] |
| Filter Tree | Filter Tree | 0.59% [1140×] | 14.4% [1285×] | 0.73% [1320×] |
| Conditional Prob. Tree (CPT) | CPT | 0.74% [41×] | 26.3% [45×] | 2.20% [115×] |
| Independent Optimization | Random Tree | 0.72% [60×] | 21.3% [59×] | 1.35% [61×] |
| Independent Optimization | Learnt Label Tree | 1.25% [60×] | 27.1% [59×] | 5.95% [61×] |
| Tree Loss Optimization | Learnt Label Tree | 2.37% [60×] | 39.6% [59×] | 10.6% [61×] |

Table 3: **Label Embeddings and Label Embedding Tree Results**

| Classifier | Tree Type | ImageNet | | | Product Images | | |
|---|---|---|---|---|---|---|---|
| | | Accuracy | Speed | Memory | Accuracy | Speed | Memory |
| One-vs-Rest | None (flat) | 2.27% | 1× | 1.2 GB | 12.6% | 1× | 170 MB |
| Compressed Sensing | None (flat) | 0.6% | 3× | 18 MB | 2.27% | 10× | 20 MB |
| Seq. Convex Embedding | None (flat) | 2.23% | 3× | 18 MB | 3.9% | 10× | 20 MB |
| Non-Convex Embedding | None (flat) | 2.40% | 3× | 18 MB | 14.1% | 10× | 20 MB |
| Label Embedding Tree | Label Tree | 2.54% | 85× | 18 MB | 13.3% | 142× | 20 MB |

**Embedding and Embedding Tree Approaches** In Table 3 we compare several label embedding methods: (i) the convex and non-convex methods from Section 5; (ii) compressed sensing; and (iii) the label embedding tree from Section 3.2. In all cases we fixed the embedding dimension $d_e = 100$. The results show that the random embeddings given by compressed sensing are inferior to learnt embeddings and Non-Convex Embedding is superior to Sequential Convex Embedding, presumably as the overall loss which is dependent on both $W$ and $V$ is jointly optimized. The latter gives results as good or superior to One-vs-Rest with modest computational gain (3× or 10× speed-up). Note, we do not detail results on the product descriptions task because no speed-up is gained there from embedding as the sparsity is already so high, however the methods still gave good test accuracy (e.g. Non-Convex Embedding yields 38.2%, which should be compared to the methods in Table 2). Finally, combining embedding and label tree learning using the "Label Embedding Tree" of Section 3.2 yields our best method on ImageNet and Product Images with a speed-up of 85× or 142× respectively with accuracy as good or better than any other method tested. Moreover, memory usage of this method (and other embedding methods) is significantly less than One-vs-Rest.

# 6  Conclusion

We have introduced an approach for fast multi-class classification by learning label embedding trees by (approximately) optimizing the overall tree loss. Our approach obtained orders of magnitude speedup compared to One-vs-Rest while yielding as good or better accuracy, and outperformed other tree-based or embedding approaches. Our method makes real-time inference feasible for very large multi-class tasks such as web advertising, document categorization and image annotation.

# Acknowledgements

We thank Ameesh Makadia for very useful discussions.

# References

[1] E. Allwein, R. Schapire, and Y. Singer. Reducing multiclass to binary: a unifying approach for margin classifiers. *Journal of Machine Learning Research (JMLR)*, 1:113–141, 2001.

[2] Y. Amit, M. Fink, N. Srebro, and S. Ullman. Uncovering shared structures in multiclass classification. In *Proceedings of the 24th international conference on Machine learning*, page 24. ACM, 2007.

[3] B. Bai, J. Weston, D. Grangier, R. Collobert, C. Cortes, and M. Mohri. Half transductive ranking. In *Artificial Intelligence and Statistics (AISTATS)*, 2010.

[4] M. Belkin and P. Niyogi. Laplacian eigenmaps and spectral techniques for embedding and clustering. *Advances in neural information processing systems*, 1:585–592, 2002.

[5] J.L. Bentley. Multidimensional binary search trees used for associative searching. *Communications of the ACM*, 18(9):517, 1975.

[6] A. Beygelzimer, J. Langford, Y. Lifshits, G. Sorkin, and A. Strehl. Conditional probability tree estimation analysis and algorithm. In *Conference in Uncertainty in Artificial Intelligence (UAI)*, 2009.

[7] A. Beygelzimer, J. Langford, and P. Ravikumar. Error-correcting tournaments. In *International Conference on Algorithmic Learning Theory (ALT)*, pages 247–262, 2009.

[8] Léon Bottou. Stochastic learning. In Olivier Bousquet and Ulrike von Luxburg, editors, *Advanced Lectures on Machine Learning*, Lecture Notes in Artificial Intelligence, LNAI 3176, pages 146–168. Springer Verlag, Berlin, 2004.

[9] K. Crammer, O. Dekel, J. Keshet, S. Shalev-Shwartz, and Y. Singer. Online passive-aggressive algorithms. *Journal of Machine Learning Research*, 7:551–585, 2006.

[10] K. Crammer and Y. Singer. On the algorithmic implementation of multiclass kernel-based vector machines. *Journal of Machine Learning Research (JMLR)*, 2:265–292, 2002.

[11] O. Dekel and O. Shamir. Multiclass-Multilabel Learning when the Label Set Grows with the Number of Examples. In *Artificial Intelligence and Statistics (AISTATS)*, 2010.

[12] J. Deng, W. Dong, R. Socher, Li-Jia Li, K. Li, and Fei-Fei Li. Imagenet: A large-scale hierarchical image database. In *Conference on Computer Vision and Pattern Recognition (CVPR)*, pages 248–255, 2009.

[13] T. Dietterich and G. Bakiri. On the algorithmic implementation of multiclass kernel-based vector machines. *Journal of Artificial Intelligence Research (JAIR)*, 2:263–286, 1995.

[14] C. Fellbaum, editor. *WordNet: An Electronic Lexical Database*. MIT Press, 1998.

[15] David Grangier and Samy Bengio. A discriminative kernel-based model to rank images from text queries. *Transactions on Pattern Analysis and Machine Intelligence*, 30(8):1371–1384, 2008.

[16] T. Hastie and R. Tibshirani. Classication by pairwise coupling. *The Annals of Statistics*, 26(2):451–471, 2001.

[17] D. Hsu, S. Kakade, J. Langford, and T. Zhang. Multi-label prediction via compressed sensing. In *Neural Information Processing Systems (NIPS)*, 2009.

[18] M.I. Jordan and R.A. Jacobs. Hierarchical mixtures of experts and the EM algorithm. *Neural computation*, 6(2):181–214, 1994.

[19] J. Langford and A. Beygelzimer. Sensitive error correcting output codes. In *Conference on Learning Theory (COLT)*, pages 158–172, 2005.

[20] U. Luxburg. A tutorial on spectral clustering. *Statistics and Computing*, 17(4):416, 2007.

[21] A.Y. Ng, M.I. Jordan, and Y. Weiss. On spectral clustering: Analysis and an algorithm. *Advances in neural information processing systems*, 2:849–856, 2002.

[22] T. Oates and D. Jensen. The effects of training set size on decision tree complexity. In *International Conference on Machine Learning (ICML)*, pages 254–262, 1997.

[23] J. Platt, N. Cristianini, and J. Shawe-Taylor. Large margin dags for multiclass classification. In *NIPS*, pages 547–553, 2000.

[24] J. Quinlan. *C4.5 : programs for machine learning*. Morgan Kaufmann, 1993.

[25] R. Rifkin and A. Klautau. In defense of one-vs-all classification. *Journal of Machine Learning Research (JMLR)*, 5:101–141, 2004.

[26] K. Weinberger and O. Chapelle. Large margin taxonomy embedding for document categorization. In *NIPS*, pages 1737–1744, 2009.

[27] P.N. Yianilos. Data structures and algorithms for nearest neighbor search in general metric spaces. In *Proceedings of the fourth annual ACM-SIAM Symposium on Discrete algorithms*, page 321. Society for Industrial and Applied Mathematics, 1993.

